# Game theoretic algorithms for Protein-DNA binding

**Luis Pérez-Breva**
CSAIL-MIT
lpbreva@csail.mit.edu

**Luis E. Ortiz**
CSAIL - MIT
leortiz@csail.mit.edu

**Chen-Hsiang Yeang**
UCSC
chyeang@soe.ucsc.edu

**Tommi Jaakkola**
CSAIL - MIT
tommi@csail.mit.edu

## Abstract

We develop and analyze game-theoretic algorithms for predicting coordinate binding of multiple DNA binding regulators. The allocation of proteins to local neighborhoods and to sites is carried out with resource constraints while explicating competing and coordinate binding relations among proteins with affinity to the site or region. The focus of this paper is on mathematical foundations of the approach. We also briefly demonstrate the approach in the context of the $\lambda$-phage switch.

## 1 Introduction

Transcriptional control relies in part on coordinate operation of DNA binding regulators and their interactions with various co-factors. We believe game theory and economic models provide an appropriate modeling framework for understanding interacting regulatory processes. In particular, the problem of understanding coordinate binding of regulatory proteins has many game theoretic properties. Resource constraints, for example, are critical to understanding who binds where. At low nuclear concentrations, regulatory proteins may occupy only high affinity sites, while filling weaker sites with increasing concentration. Overlapping or close binding sites create explicit competition for the sites, the resolution of which is guided by the available concentrations around the binding sites. Similarly, explicit coordination such as formation of larger protein complexes may be required for binding or, alternatively, binding may be facilitated by the presence of another protein. The key advantage of games as models of binding is that they can provide causally meaningful predictions (binding arrangements) in response to various experimental perturbations or disruptions.

Our approach deviates from an already substantial body of computational methods used for resolving transcriptional regulation (see, e.g., [3, 10]). From a biological perspective our work is closest in spirit to more detailed reaction equation models [5, 1], while narrower in scope. The mathematical approach is nevertheless substantially different.

## 2 Protein-DNA binding

We decompose the binding problem into *transport* and *local binding*. By *transport*, we refer to the mechanism that transports proteins to the neighborhood of sites to which they have affinity. The biological processes underlying the transport are not well-understood although several hypotheses exist[12, 4]. We abstract the process initially by assuming separate affinities for proteins to explore neighborhoods of specific sites, modulated by whether the sites are available. This abstraction does not address the dynamics of the transport process and therefore does not distinguish (nor stand in contradiction to) underlying mechanisms that may or may not involve diffusion as a major com-

ponent. We aim to capture the differentiated manner in which proteins may accumulate in the neighborhoods of sites depending on the overall nuclear concentrations and regardless of the time involved.

*Local binding*, on the other hand, captures which proteins bind to each site as a consequence of local accumulations or concentrations around the site or a larger region. In a *steady state*, the local environment of the site is assumed to be *closed* and *well-mixed*. We therefore model the binding as being governed by chemical equilibria: for a type of protein $i$ around site $j$, {free protein i} + {free site j} $\rightleftharpoons$ {bound ij}, where concentrations involving the site should be thought of as time averages or averages across a population of cells depending on the type of predictions sought. The concentrations of various molecular species around and bound to the sites as well as the rate at which the sites are occupied are then governed by the law of mass action at chemical equilibrium: [bound $ij$]/([free protein $i$][free site $j$]) $= K_{ij}$, where $i$ ranges over proteins with affinity to site $j$ and $K_{ij}$ is a positive equilibrium constant characterizing protein $i$'s ability to bind to site $j$ in the absence of other proteins.

Broadly speaking, the combination of transport and local binding results in an arrangement of proteins along the possible DNA binding sites. This is what we aim to predict with our game-theoretic models, not how such arrangements are reached. The predictions should be viewed as functions of the overall (nuclear) concentrations of proteins, the affinities of proteins to explore neighborhoods of individual sites, as well as the equilibrium constants characterizing the ability of proteins to bind to specific sites when in close proximity. Any perturbation of such parameters leads to a potentially different arrangement that we can predict.

## 3  Game Theoretic formulation

There are two types of players in our game, proteins and sites. A *protein-player* refers to a type of protein, not an individual protein, and decides how its nuclear concentration is allocated to the proximity of sites (transport process). The protein-players are assumed non-cooperative and rational. In other words, their allocations are based on the transport affinities and the availability of sites rather than through some negotiation process involving multiple proteins. The non-coopeative nature of the protein allocations does not, however, preclude the formation of protein complexes or binding facilitated by other proteins. Such extensions can be incorporated at the sites.

Each possible binding site is associated with a *site-player*. Site-players choose the fraction of time (or fraction of cells in a population) a specific type of protein is bound to the site. The site may also remain empty. The strategies of the site-players are guided by local chemical equilibria. Indeed, the site-players are introduced merely to reproduce this physical understanding of the binding process in a game theoretic context. The site-players are non-cooperative and self-interested, always aiming and succeeding at reproducing the local chemical equilibria.

The binding game has no global objective function that serves to guide how the players choose their strategies. The players choices are instead guided by their own utilities that depend on the choices of other players. For example, the protein-player allocates its nuclear concentration to the proximity of the sites based on how occupied the sites are, i.e., in a manner that depends on the strategies of the site-players. Similarly, the site-players reproduce the chemical equilibrium at the sites on the basis of the available local protein concentrations, i.e., depending on the choices of the protein-players.

The predictions we can make based on the game theoretic formulation are *equilibria of the game* (not to be confused with the local chemical equilibria at the sites). At an equilibrium, no reallocation of proteins to sites is required and, conversely, the sites have reproduced the local chemical equilibria based on the current allocations of proteins. While games need not have equilibria in pure strategies (actions available to the players), our game will always have one.

## 4  The binding game

To specify the game more formally we proceed to define players' strategies, their utilities, and the notion of an equilibrium of the game. To this end, let $f^i$ represent the (nuclear) concentration of protein $i$. This is the amount of protein available to be allocated to the neighborhoods of sites. The fraction of protein $i$ allocated to site $j$ is specified by $p^i_j$, where $\sum_j p^i_j = 1$. The numerical values

of $p_j^i$, where $j$ ranges over the possible sites, define a possible strategy for the $i^{th}$ protein player. The set of such strategies is denoted by $\mathcal{P}^i$. The choices of which strategies to play are guided by parameters $E_{ij}$, the affinity of protein $i$ to explore the neighborhood of site $j$ (we will generally index proteins with $i$ and sites with $j$). The utility for protein $i$, defined below, provides a numerical ranking of possible strategy choices and is parameterized by $E_{ij}$. Each player aims to maximize its own utility over the set of possible strategy choices.

The strategy for site-player $j$ specifies the fraction of time that each type of protein is actually bound to the site. The strategy is denoted by $s_i^j$, where $i$ ranges over proteins with affinity to the site. Note that the values of $s_i^j$ are in principle observable from binding assays (cf. [9]). $\sum_i s_i^j \leq 1$ since there is only one site and it may remain empty part of the time. The availability of site $j$ is $1 - \sum_i s_i^j \leq 1$, i.e., the fraction of time that nothing is bound. We will also use $\alpha^j = \sum_i s_i^j$ to denote how occupied the site is. The utilities of the site players will depend on $K_{ij}$, the chemical equilibrium constants characterizing the local binding reaction between protein $i$ and site $j$.

**Utilities** The utility function for protein-player $i$ is formally defined as

$$u_i(p^i, s) \equiv \sum_j p_j^i E_{ij} (1 - \sum_{i'} s_{i'}^j) + \beta H(p^i) \tag{1}$$

where $H(p^i) = -\sum_j p_j^i \log p_j^i$ is the Shannon entropy of the strategy $p_j^i$ and $j$ ranges over possible sites. The utility of the protein-player essentially states that protein $i$ "prefers" to be around sites that are unbound and for which it has high affinity. The parameter $\beta \geq 0$ balances how much protein allocations are guided by the differentiated process, characterized by the exploration affinities $E_{ij}$, as opposed to allocated uniformly (maximizing the entropy function). Since the overall scaling of the utilities is immaterial, only the ratios $E_{ij}/\beta$ are relevant for guiding the protein-players. Note that since the utility depends on the strategies of site-players through $(1 - \sum_{i'} s_{i'}^j)$, one cannot find the equilibrium strategy for proteins by considering $s_i^j$ to be fixed; the sites will respond to any $p_j^i$ chosen by the protein-player.

As discussed earlier, the site-players always reproduce the chemical equilibrium between the site and the protein species allocated to the neighborhood of the site. The utility for site-player $i$ is defined such that the maximizing strategy corresponds to the chemical equilibrium:

$$s_i^j / \left[ (p_j^i f^i - s_i^j)(1 - \sum_{i'} s_{i'}^j) \right] = K_{ij} \tag{2}$$

where $s_i^j$ specifies how much protein $i$ is bound, the first term in the denominator $(p_j^i f^i - s_i^j)$ specifies the amount of free protein $i$, and the second term $(1 - \sum_{i'} s_{i'}^j)$, the fraction of time the site is available. The equilibrium equation holds for all protein species around the site and for the same strategy $\{s_i^j\}$ of the site-player. The units of each "concentration" in the above equation should be interpreted as numbers of available molecules (e.g., there's only one site). The utility function that reproduces this chemical equilibrium when maximized over possible strategies is given by

$$v_j(s^j, p) \equiv \sum_i s_i^j - K_{ij}(p_j^i f^i - s_i^j)\left(1 - \sum_{i'} s_{i'}^j\right) \tag{3}$$

subject to $s_i^j \leq K_{ij}(p_j^i f^i - s_i^j)(1 - \sum_{i'} s_{i'}^j)$, $s_i^j \leq p_j^i f^i$, and $\sum_{i'} s_{i'}^j \leq 1$. These constraints guarantee that the utility is always non-positive and zero exactly when the chemical equilibrium holds. $s_i^j \leq p_j^i f^i$ ensures that we cannot have more protein bound than is allocated to the proximity of the site. These constraints define the set of strategies available for site-player $j$ or $\mathcal{S}^j(p)$. Note that the available strategies for the site-player depend on the current strategies for protein-players. The set of strategies $\mathcal{S}^j(p)$ is not convex.

## 4.1 The game and equilibria

The *protein-DNA binding game* is now fully specified by the set of parameters $\{E_{ij}/\beta\}$, $\{K_{ij}\}$ and $\{f^i\}$, along with the utility functions $\{u_i\}$ and $\{v_j\}$ and the allocation constraints $\{\mathcal{P}^i\}$ and $\{\mathcal{S}^j\}$.

We assume that the biological system being modeled reaches a steady state, at least momentarily, preserving the average allocations. In terms of our game theoretic model, this corresponds to what

we call an *equilibrium* of the game. Informally, an equilibrium of a game is a strategy for each player such that no individual has any incentive to unilaterally deviate from their strategy. Formally, if the allocations $(\bar{p}, \bar{s})$ are such that for each protein $i$ and each site $j$,

$$\bar{p}^i \in \arg\max_{p^i \in \mathcal{P}^i} u_i(p^i, \bar{s}), \text{ and } \bar{s}^j \in \arg\max_{s^j \in \mathcal{S}^j(\bar{p}_j)} v_j(s^j, \bar{p}_j), \tag{4}$$

then we call $(\bar{p}, \bar{s})$ an *equilibrium* of the protein-DNA binding game. Put another way, at an equilibrium, the current strategies of the players must be among the strategies that maximize their utilities assuming the strategies of other players are held fixed.

Does the protein-DNA binding game always have an equilibrium? While we have already stated this in the affirmative, we emphasize that there is no reason *a priori* to believe that there exists an equilibrium in the pure strategies, especially since the sets of possible strategies for the site-players are non-convex (cf. [2]). The existence is guaranteed by the following theorem:

**Theorem 1.** *Every protein-DNA binding game has an equilibrium.*

A constructive proof is provided by the algorithm discussed below. The theorem guarantees that at least one equilibrium exists but there may be more than one. At any such equilibrium of the game, all the protein species around each site are at a chemical equilibrium; that is, if $(\bar{p}, \bar{s})$ is an equilibrium of the game, then for all sites $j$ and proteins $i$, $\bar{s}^j$ and $\bar{p}_j^i$ satisfy (2). Consequently, the site utilities $v_j(\bar{s}^j, \bar{p}_j)$ are all zero for the equilibrium strategies.

## 4.2 Computing equilibria

The equilibria of the binding game represent predicted binding arrangements. Our game has special structure and properties that permit us to find an equilibrium efficiently through a simple iterative algorithm. The algorithm monotonically fills the sites up to the equilibrium levels, starting with all sites empty.

We begin by first expressing any joint equilibrium strategy of the game as a function of how filled the sites are, and reduce the problem of finding equilibria to finding fixed points of a monotone function. To this end, let $\alpha^j = \sum_{i'} s_{i'}^j$ denote site $j$ occupancy, the fraction of time it is bound by any protein. $\alpha^j$'s are real numbers in the interval $[0, 1]$. If we fix $\alpha = (\alpha^1, \dots, \alpha^m)$, i.e., the occupancies for all the $m$ sites, then we can readily obtain the maximizing strategies for proteins expressed as a function of site occupancies: $p_j^i(\alpha) \propto \exp(E_{ij}(1 - \alpha^j)/\beta)$, where the maximizing strategies are functions of $\alpha$. Similarly, at the equilibrium, each site-player achieves a local chemical equilibrium specified in (2). By replacing $\alpha^j = \sum_{i'} s_{i'}^j$, and solving for $s_i^j$ in (2), we get

$$s_i^j(\alpha) = \frac{K_{ij}(1 - \alpha^j)}{1 + K_{ij}(1 - \alpha^j)} p_j^i(\alpha) f^i \tag{5}$$

So, for example, the fraction of time the site is bound by a specific protein is proportional to the amount of that protein in the neighborhood of the site, modulated by the equilibrium constant. Note that $s_i^j(\alpha)$ depends not only on how filled site $j$ is but also on how occupied the other sites are through $p_j^i(\alpha)$.

The equilibrium condition can be now expressed solely in terms of $\alpha$ and reduces to a simple consistency constraint: overall occupancy should equal the fraction of time any protein is bound or

$$\alpha^j = \sum_i s_i^j(\alpha) = \sum_i \frac{K_{ij}(1 - \alpha^j)}{1 + K_{ij}(1 - \alpha^j)} p_j^i(\alpha) f^i = G^j(\alpha) \tag{6}$$

We have therefore reduced the problem of finding equilibria of the game to finding fixed points of the mapping $G^j(\alpha) = \sum_i s_i^j(\alpha)$. This mapping, written explicitly as has a simple but powerful monotonicity property that forms the basis for our iterative algorithm. Specifically,

**Lemma 1.** *Let $\alpha^{-j}$ denote all components $\alpha^k$ except $\alpha^j$. Then for each $j$, $G^j(\alpha) \equiv G^j(\alpha^j, \alpha^{-j})$ is a strictly decreasing function of $\alpha^j$ for any fixed $\alpha^{-j}$.*

We omit the proof as it is straightforward. This lemma, together with the fact that $G^j(1, \alpha^{-j}) = 0$, immediately guarantees that there is a *unique* solution to $\alpha^j = G^j(\alpha^j, \alpha^{-j})$ for any fixed and valid $\alpha^{-j}$. The solution $\alpha^j$ also lies in the interval $[0, 1]$ and can be found efficiently via binary search.

**The algorithm** Let $\alpha(t)$ denote the site occupancies at the $t^{th}$ iteration of the algorithm. $\alpha^j(t)$ specifies the $j^{th}$ component of this vector, while $\alpha^{-j}(t)$ contains all but the $j^{th}$ component. The algorithm proceeds as follows:

- Set $\alpha^j(0) = 0$ for all $j = 1, \ldots, m$.
- Find each new component $\alpha^j(t+1)$, $j = 1, \ldots, m$, on the basis of the corresponding $\alpha^{-j}(t)$ such that $\alpha^j(t+1) = G^j(\alpha^j(t+1), \alpha^{-j}(t))$
- Stop when $\alpha^j(t+1) \approx \alpha^j(t)$ for all $j = 1, \ldots, m$.

Note that the inner loop of the algorithm, i.e., finding $\alpha^j(t+1)$ on the basis of $\alpha^{-j}(t)$ reduces to a simple binary search as discussed earlier. The algorithm generates a monotonically increasing sequence of $\alpha$'s that converge to a fixed point (equilibrium) solution.

We also provide a formal convergence analysis of the algorithm. To this end, we begin with the following critical lemma.

**Lemma 2.** *Let $\alpha_1$ and $\alpha_2$ be two possible assignments to $\alpha$. If for all $k \neq j$, $\alpha_1^k \leq \alpha_2^k$, then $G^j(\alpha^j, \alpha_1^{-j}) \leq G^j(\alpha^j, \alpha_2^{-j})$ for all $\alpha^j$.*

The proof is straightforward and essentially based on the fact that $\alpha_1^{-j}$ and $\alpha_2^{-j}$ appear only in the normalization terms for the protein allocations. We omit further details for brevity. On the basis of this lemma, we can show that the algorithm indeed generates a monotonically increasing sequence of $\alpha$'s

**Theorem 2.** $\alpha^j(t+1) \geq \alpha^j(t)$ *for all $j$ and $t$.*

*Proof.* By induction. Since $\alpha^j(0) = 0$ and the range of $G^j(\alpha^j, \alpha^{-j}(0))$ lies in $[0, 1]$, clearly $\alpha^j(1) \geq \alpha^j(0)$ for all $j$. Assume then that $\alpha^j(t) \geq \alpha^j(t-1)$ for all $j$. We extend the induction step by contradiction. Suppose $\alpha^j(t+1) < \alpha^j(t)$ for some $j$. Then

$$\begin{aligned} \alpha^j(t+1) < \alpha^j(t) &= G^j(\alpha^j(t), \alpha^{-j}(t-1)) \leq G^j(\alpha^j(t), \alpha^{-j}(t)) \\ &< G^j(\alpha^j(t+1), \alpha^{-j}(t)) = \alpha^j(t+1) \end{aligned}$$

which is a contradiction. The first "$\leq$" follows from the induction hypothesis and lemma 2, and the last "$<$" derives from lemma 1 and $\alpha^j(t+1) < \alpha^j(t)$. $\square$

Since $\alpha^j(t)$ for any $t$ will always lie in the interval $[0, 1]$, and because of the continuity of $G^j(\alpha^j, \alpha^{-j})$ in the two arguments, the algorithm is guaranteed to converge to a fixed point solution. More formally, the Monotone Convergence Theorem for sequences and the continuity of $G^j$'s imply that

**Theorem 3.** *The algorithm converges to a fixed point $\bar{\alpha}$ such that $\bar{\alpha}^j = G^j(\bar{\alpha}^j, \bar{\alpha}^{-j})$ for all $j$.*

### 4.3 The $\lambda$-phage binding game

We use the well-known $\lambda$-phage viral infection [11, 1] to illustrate the game theoretic approach. A genetic two-state control switch specifies whether the infection remains dormant (lysogeny) or whether the viral DNA is aggressively replicated (lysis). The components of the $\lambda-$switch are 1) two adjacent genes *cI* and *Cro* that encode $cI_2$ and *Cro* proteins, respectively; 2) the promoter regions $P_{RM}$ and $P_R$ of these genes, and 3) an operator ($O_R$) with three binding sites $O_R1$, $O_R2$, and $O_R3$. We focus on lysogeny, in which $cI_2$ dominates over Cro. There are two relevant protein-players, RNA-polymerase and $cI_2$, and three sites, $O_R1$, $O_R2$, and $O_R3$ (arranged close together in this order). Since the presence of $cI_2$ in either $O_R1$ or $O_R3$ blocks the access of RNA-polymerase to the promoter region $P_R$, or $P_{RM}$ respectively, we can safely restrict ourselves to operator sites as the site-players. There are three phases of operation depending on the concentration of $cI_2$:

1. $cI_2$ binds to $O_R1$ first and blocks the Cro promoter $P_R$
2. Slightly higher concentrations of $cI_2$ lead to binding at $O_R2$ which in turn facilitates RNA-polymerase to initiate transcription at $P_{RM}$
3. At sufficiently high levels $cI_2$ also binds to $O_R3$ and inhibits its own transcription

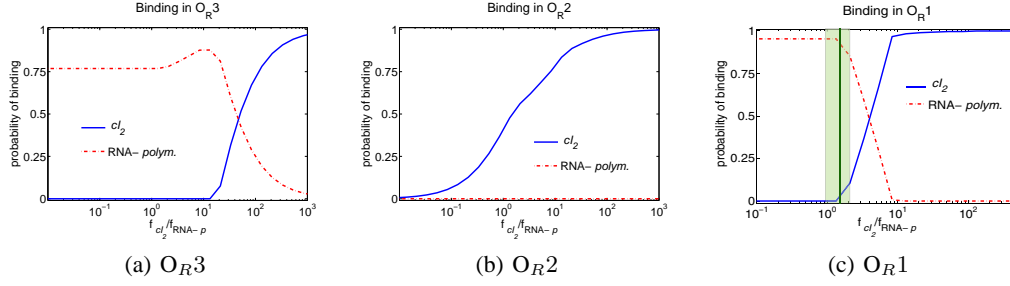

|  (a) $O_R3$ | (b) $O_R2$ | (c) $O_R1$ |

Figure 1: Predicted protein binding to sites $O_R3$, $O_R2$, and $O_R1$ for increasing amounts of $cI_2$. The rightmost figure illustrates a comparison with [1]. The shaded area indicates the range of concentrations of $cI_2$ at which stochastic simulation predicts a decline in transcription from $O_R1$. Our model predicts that $cI_2$ begins to occupy $O_R1$ at the same concentration.

**Game parameters** The game requires three sets of parameters: chemical equilibrium constants, affinities, and protein concentrations. To use constants derived from experiment we assign units to these quantities. We define $f^i$ as the total number of proteins $i$ available, and arrange the units of $K_{ij}$ accordingly:

$$f^i \equiv \widetilde{f}^i \, V_T N_A, \qquad\qquad K_{ij} \equiv \widetilde{K}_{ij}/(N_A V_S) \qquad\qquad \widetilde{K}_{ij} = e^{-\Delta G/RT} \qquad (7)$$

where $V_T$ and $V_S$ are the volumes of cell and site neighborhood, respectively, $N_A$ is the Avogadro number, $R$ is the universal gas constant, $T$ is temperature, $\widetilde{f}^i$ is the concentration of protein $i$ in the cell, and $\widetilde{K}_{ij}$ is the equilibrium constant in units of $\ell/mol$. As we show in [6] these definitions are consistent with our previous derivation. Note that when game parameters are learned from data any dependence on the volumes will be implicit. For a typical *Escherichia coli* ( $2\mu m$ length) at room temperature, the Gibbs' Free energies $\Delta G$ tabulated by [11] yield the equilibrium constants shown below; in addition, we set transport affinities in accordance with the qualitative description in [7, 8],

| $K_{ij}$ | $O_R3$ | $O_R2$ | $O_R1$ |       | $E_{ij}$ | $O_R3$ | $O_R2$ | $O_R1$ |
|----------|--------|--------|--------|-------|----------|--------|--------|--------|
| $cI_2$   | .0020  | .0020  | .0296  |       | $cI_2$   | .1     | .1     | 1      |
| RNA-p    | .0212  | 0      | .1134  |       | RNA-p    | .2     | .01    | 1      |

Note that the overall scaling of the affinities is immaterial; only their relative values will guide the protein-players. Note also that we have chosen not to incorporate any protein-protein interactions in the affinities.

Finally, we set $\widetilde{f}_{RNA-p} = 30nM$ (cf. [11]) (around $f_{RNA-p} \simeq 340$ copies for a typical *E. coli*). And varied $f_{cI_2}$ from 1 to $10,000$ copies to study the dynamical behavior of the lysogeny cycle. The results are reported as a function of the ratio $f_{cI_2}/f_{RNA-p}$. We set $\beta = 10^{-5}$.

**Simulation Results** The predictions from the game theoretic model exactly mirror the known behavior. Here we summarize the main results and refer the reader to [6] for a thorough analysis.

Figure 1 illustrates how the binding at different sites changes as a function of increasing $f_{cI_2}$. The simulation mirrors the behavior of the lysogeny cycle discussed earlier. Although our model does not capture dynamics, and figure 1 does not involve time, it is nevertheless useful for assessing quantitative changes and the order of events as a function of increasing $f_{cI_2}$. Note, for example, that the levels at which $cI_2$ occupies $O_R1$ and $O_R2$ rise much faster than at $O_R3$. While the result is expected, the behavior is attributed to protein-protein interactions which are not encoded in our model. Similarly, RNA-polymerase occupation at $O_R3$ bumps up as the probability that $O_R2$ is bound by $cI_2$ increases. In [6] we further discuss the implications of the simultaneous occupancy of $O_R1$ and $O_R2$, via simulation of $O_R1$ knockout experiments.

Finally, figure 1(c) shows a comparison with stochastic simulation (*v.* [1]). Our model predicts that $cI_2$ begins binding $O_R1$ at the same level as [1] predicts a decline in the transcription of Cro. While consistent, we emphasize that the methods differ in their goals; stochastic simulation focuses on the dynamics of transcription while we study the strategic allocation of proteins as a function of their concentration.

## 4.4   A structured extension

The game theoretic formulation of the binding problem described previously involves a transport mechanism that is specific to individual sites. In other words, proteins are allocated to the proximity of sites based on parameters $E_{ij}$ and occupancies $\alpha^j$ associated with individual sites. We generalize the game further here by assuming that the transport mechanism has a coarser spatial structure, e.g., specific to promoters (regulatory regions of genes) rather than sites. In this extension the amount of protein allocated to any promoter is shared by the sites it contains. The sharing creates specific challenges to the algorithms for finding the equilibria and we will address those challenges here.

Let $\mathcal{R}$ represent possible promoter regions each of which may be bound by multiple proteins (at distinct or overlapping sites). Let $p^i = \{p_r^i\}_{r \in \mathcal{R}}$ represent an allocation of protein $i$ into these regions in a manner that is not specific to the possible sites within each promoter. The utility for protein $i$ is given by

$$u_i(p^i) = \sum_{r \in \mathcal{R}} p_r^i \, E_{ir}(a^r) + \beta H(p^i)$$

where $N(r)$ is the set of possible binding sites within promoter region $r$ and $a^r = \sum_{j \in N(r)} \alpha^j$ is the overall occupancy of the promoter (how many proteins bound). As before, $\alpha^j = \sum_{i \in P} s_i^j$, where the summation is over proteins. $N(r) \cap N(r') = \emptyset$ whenever $r \neq r'$ (promoters don't share sites). We assume only that $E_{ir}(a^r)$ is a decreasing and a differentiable function of $a^r$. The protein utility is based on the assumption that the attraction to the promoter decreases based on the number of proteins already bound at the promoter. The maximizing strategy for protein $i$ given $a^r = \sum_{j \in N(r)} \alpha^j$ for all $r$, is $p_r^i(a) \propto \exp(E_{ir}(a^r)/\beta)$, where $a = \{a^r\}_{r \in \mathcal{R}}$.

Sites $j \in N(r)$ within a promoter region $r$ reproduce the following chemical equilibrium

$$s_i^j \big/ \Big[ (f^i p_r^i(a) - \textstyle\sum_{k \in N(r)} s_i^k)(1 - \alpha^j) \Big] = K_{ij}$$

for all proteins $i \in P$. Note the shared protein resource within the promoter. We can find this chemical equilibrium by solving the following fixed point equations

$$\alpha^j = \sum_{i \in P} \frac{K_{ij}(1 - \alpha^j)}{1 + \sum_{k \in N(r)} K_{ik}(1 - \alpha^k)} \, f^i p_r^i(a) = G_r^j(\alpha, a^{-r})$$

The site occupancies $\alpha^j$ are now tied within the promoter as well as influencing the overall allocation of proteins across different promoters through $a = \{a^r\}_{r \in \mathcal{R}}$. The following theorem provides the basis for solving the coupled fixed point equations:

**Theorem 4.** *Let $\{\hat{\alpha}_1^j\}$ be the fixed point solution $\alpha_1^j = G_r^j(\alpha_1, a_1^{-r})$ and $\{\hat{\alpha}_2^j\}$ the solution to $\alpha_2^j = G_r^j(\alpha_2, a_2^{-r})$. If $a_1^l \leq a_2^l$ for all $l \neq r$ then $\hat{a}_1^r \leq \hat{a}_2^r$.*

The proof is not straightforward but we omit it for brevity (two pages). The result guarantees that if we can solve the fixed point equations within each promoter then the overall occupancies $\{a^r\}_{r \in \mathcal{R}}$ have the same monotonicity property as in the simpler version of the game where $a^r$ consisted of a single site. In other words, any algorithm that successively solves the fixed point equations within promoters will result in a monotone and therefore convergent filling of the promoters, beginning with all empty promoters.

We will redefine the notation slightly to illustrate the algorithm for finding the solution $\alpha^j = G_r^j(\alpha, a^{-r})$ for $j \in N(r)$ where $a^{-r}$ is fixed. Specifically, let

$$G_r^j(\alpha^j, \underline{\alpha}^{-j}, \bar{\alpha}^j, a^{-r}) = \sum_{i \in P} \frac{K_{ij}(1 - \alpha^j)}{1 + K_{ij}(1 - \alpha^j) + \sum_{k \neq j} K_{ik}(1 - \underline{\alpha}^k)} \, f^i p_r^i(\alpha^j, \bar{\alpha}^{-j}, a^{-r})$$

In other words, the first argument refers to $\alpha^j$ anywhere on the right hand side, the second argument refers to $\alpha^{-j}$ in the denominator of the first expression in the sum, and the third argument refers to $\alpha^{-j}$ in $p_r^i(\cdot)$. The algorithm is now defined as follows: initialize by setting $\underline{\alpha}^j(0) = 0$ and $\bar{\alpha}^j(0) = 1$ for all $j \in N(r)$, then

> *Iteration t, upper bounds:* Find $\hat{\alpha}^j = G_r^j(\hat{\alpha}^j, \bar{\alpha}^{-j}(t), \underline{\alpha}^{-j}(t), a^{-r})$ separately for each $j \in N(t)$. Update $\bar{\alpha}^j(t+1) = \hat{\alpha}^j, j \in N(r)$

*Iteration t, lower bounds:* Find $\hat{\alpha}^j = G_r^j(\hat{\alpha}^j, \underline{\alpha}^{-j}(t), \bar{\alpha}^{-j}(t+1), a^{-r})$ separately for each $j \in N(r)$. Update $\underline{\alpha}^j(t+1) = \hat{\alpha}^j$, $j \in N(r)$

The iterative optimization proceeds until[1] $\bar{\alpha}^j(t) - \underline{\alpha}^j(t) \leq \epsilon$ for all $j \in N(r)$. The algorithm successively narrows down the gap between upper and lower bounds. Specifically, $\bar{\alpha}^j(t+1) \leq \bar{\alpha}^j(t)$ and $\underline{\alpha}^j(t+1) \geq \underline{\alpha}^j(t)$. The fact that these indeed remain upper and lower bounds follows directly from the fact that $G_r^j(\cdot, \underline{\alpha}^{-j}, \bar{\alpha}^j, a^{-r})$, viewed as a function of the first argument, increases uniformly as we increase the components of the second argument. Similarly, it uniformly decreases as a function of the third argument.

## 5   Discussion

We have presented a game theoretic approach to predicting protein arrangements along the DNA. The model is complete with convergent algorithms for finding equilibria on a genome-wide scale. The results from the small scale application are encouraging. Our model successfully reproduces known behavior of the $\lambda-$switch on the basis of molecular level competition and resource constraints, without the need to assume protein-protein interactions between $cI_2$ dimers and $cI_2$ and RNA-polymerase. Even in the context of this well-known sub-system, however, few quantitative experimental results are available about binding (see the comparison). Proper validation and use of our model therefore relies on estimating the game parameters from available protein-DNA binding data. This will be addressed in subsequent work.

*This work was supported in part by NIH grant GM68762 and by NSF ITR grant 0428715. Luis Pérez-Breva is a "Fundación Rafael del Pino" Fellow.*

## Footnotes

[1]In the case of multiple equilibria the bounds might converge but leave a finite gap. The algorithm will identify those cases as the monotone convergence of the bounds can be assessed separately.

## References

[1] Adam Arkin, John Ross, and Harley H. McAdams. Stochastic kinetic analysis of developmental pathway bifurcation in phage $\lambda$-infected escherichia coli cells. *Genetics*, 149:1633–1648, August 1998.

[2] Kenneth J. Arrow and Gerard Debreu. Existence of an equilibrium for a competitive economy. *Econometrica*, 22(3):265–290, July 1954.

[3] Z. Bar-Joseph, G. Gerber, T. Lee, N. Rinaldi, J. Yoo, B. Gordon F. Robert, E. Fraenkel, T. Jaakkola, R. Young, and D. Gifford. Computational discovery of gene modules and regulatory networks. *Nature Biotechnology*, 21(11):1337–1342, 2003.

[4] Otto G. Berg, Robert B. Winter, and Peter H. von Hippel. Diffusion- driven mechanisms of protein translocation on nucleic acids. 1. models and theory. *Biochemistry*, 20(24):6929–48, November 1981.

[5] HarleyH. McAdams and Adam Arkin. Stochastic mechanisms in geneexpression. *PNAS*, 94(3):814–819, 1997.

[6] Luis Pérez-Breva, Luis Ortiz, Chen-Hsiang Yeang, and Tommi Jaakkola. DNA binding and games. Technical Report MIT-CSAIL-TR-2006-018, Massachusetts Institute of Technology, Computer Science and Artificial Intelligence Laboratory, March 2006.

[7] Mark Ptashne. *A Genetic Switch: Gene control and phage $\lambda$*. Cell Press AND Blackwell Scientific Publications, 3rd edition, 1987.

[8] Mark Ptashne and Alexander Gann. *Genes and Signals*. Cold Spring Harbor Laboratory press, 1st edition, 2002.

[9] Bing Ren, Franois Robert, John J. Wyrick, Oscar Aparicio, Ezra G. Jennings, Itamar Simon, Julia Zeitlinger, Jrg Schreiber, Nancy Hannett, Elenita Kanin, Thomas L. Volkert, Christopher J. Wilson, Stephen P. Bell, , and Richard A. Young. Genome-wide location and function of DNA-binding proteins. *Science*, 290(2306), December 2000.

[10] E. Segal, M. Shapira, A. Regev, D. Pe'er, D. Botstein, D. Koller, and N. Friedman. Module networks: identifying regulatory modules and their condition-specific regulators from gene expression data. *Nature Genetics*, 34(2):166–76, 2003.

[11] Madeline A. Shea and Gary K. Ackers. The $o_r$ control system of bacteriophage lambda. a physical-chemical model for gene regulation. *Journal of Molecular Biology*, 181:211–230, 1985.

[12] Neil P. Stanford, Mark D. Szczelkun, John F. Marko, and Stephen E. Halford. One- and three-dimensional pathways for proteins to reach specific DNA sites. *EMBO*, 19(23):6546–6557, December 2000.

